# Expectation Maximization and Posterior Constraints

**João V. Graça**
L²F INESC-ID
INESC-ID
Lisboa, Portugal

**Kuzman Ganchev**
Computer & Information Science
University of Pennsylvania
Philadelphia, PA

**Ben Taskar**
Computer & Information Science
University of Pennsylvania
Philadelphia, PA

## Abstract

The expectation maximization (EM) algorithm is a widely used maximum likelihood estimation procedure for statistical models when the values of some of the variables in the model are not observed. Very often, however, our aim is primarily to find a model that assigns values to the latent variables that have intended meaning for our data and maximizing expected likelihood only sometimes accomplishes this. Unfortunately, it is typically difficult to add even simple a-priori information about latent variables in graphical models without making the models overly complex or intractable. In this paper, we present an efficient, principled way to inject rich constraints on the posteriors of latent variables into the EM algorithm. Our method can be used to learn tractable graphical models that satisfy additional, otherwise intractable constraints. Focusing on clustering and the alignment problem for statistical machine translation, we show that simple, intuitive posterior constraints can greatly improve the performance over standard baselines and be competitive with more complex, intractable models.

## 1 Introduction

In unsupervised problems where observed data has sequential, recursive, spatial, relational, or other kinds of structure, we often employ statistical models with latent variables to tease apart the underlying dependencies and induce meaningful semantic parts. Part-of-speech and grammar induction, word and phrase alignment for statistical machine translation in natural language processing are examples of such aims. Generative models (graphical models, grammars, etc.) estimated via EM [6] are one of the primary tools for such tasks. The EM algorithm attempts to maximize the likelihood of the observed data marginalizing over the hidden variables. A pernicious problem with most models is that the data likelihood is not convex in the model parameters and EM can get stuck in local optima with very different latent variable posteriors. Another problem is that data likelihood may not guide the model towards the intended meaning for the latent variables, instead focusing on explaining irrelevant but common correlations in the data. Very indirect methods such as clever initialization and feature design (as well as ad-hoc procedural modifications) are often used to affect the posteriors of latent variables in a desired manner.

By allowing to specify prior information *directly* about posteriors of hidden variables, we can help avoid these difficulties. A somewhat similar in spirit approach is evident in work on multivariate information bottleneck [8], where extra conditional independence assumptions between latent variables can be imposed to control their "meaning". Similarly, in many semisupervised approaches, assumptions about smoothness or other properties of the posteriors are often used as regularization [18, 13, 4]. In [17], deterministic annealing was used to to explicitly control a particular feature of the posteriors of a grammar induction model. In this paper, we present an approach that effectively incorporates rich constraints on posterior distributions of a graphical model into a simple and efficient EM scheme. An important advantage of our approach is that the E-step remains tractable in a large class of problems even though incorporating the desired constraints directly into the model would make it intractable. We test our approach on synthetic clustering data as well as statistical

word alignment and show that we can significantly improve the performance of simple, tractable models, as evaluated on hand-annotated alignments for two pairs of languages, by introducing intuitive constraints such as limited fertility and the agreement of two models. Our method is attractive in its simplicity and efficiency and is competitive with more complex, intractable models.

## 2 Expectation Maximization and posterior constraints

We are interested in estimating the parameters $\theta$ of a model $p_\theta(\mathbf{x}, \mathbf{z})$ over observed variables $\mathbf{X}$ taking values $\mathbf{x} \in \mathcal{X}$ and latent variables $\mathbf{Z}$ taking values $\mathbf{z} \in \mathcal{Z}$. We are often even more interested in the induced posterior distribution over the latent variables, $p_\theta(\mathbf{z} \mid \mathbf{x})$, as we ascribe domain-specific semantics to these variables. We typically represent $p_\theta(\mathbf{x}, \mathbf{z})$ as a directed or undirected graphical model (although the discussion below also applies to context free grammars and other probabilistic models). We assume that computing the joint and the marginals is tractable and that the model factors across cliques as follows: $p_\theta(\mathbf{x}, \mathbf{z}) \propto \prod_\alpha \phi_\theta(\mathbf{x}_\alpha, \mathbf{z}_\alpha)$, where $\phi_\theta(\mathbf{x}_\alpha, \mathbf{z}_\alpha)$ are clique potentials or conditional probability distributions.

Given a sample $S = \{\mathbf{x}^1, \ldots, \mathbf{x}^n\}$, EM maximizes the average log likelihood function $L_S(\theta)$ via an auxiliary lower bound $F(q, \theta)$ (cf. [14]):

$$
\begin{aligned}
L_S(\theta) &= \mathbf{E}_S[\log p_\theta(\mathbf{x})] = \mathbf{E}_S\left[\log \sum_{\mathbf{z}} p_\theta(\mathbf{x}, \mathbf{z})\right] = \mathbf{E}_S\left[\log \sum_{\mathbf{z}} q(\mathbf{z} \mid \mathbf{x})\frac{p_\theta(\mathbf{x}, \mathbf{z})}{q(\mathbf{z} \mid \mathbf{x})}\right] \quad (1) \\
&\geq \mathbf{E}_S\left[\sum_{\mathbf{z}} q(\mathbf{z} \mid \mathbf{x}) \log \frac{p_\theta(\mathbf{x}, \mathbf{z})}{q(\mathbf{z} \mid \mathbf{x})}\right] = F(q, \theta), \quad (2)
\end{aligned}
$$

where $\mathbf{E}_S[f(\mathbf{x})] = \frac{1}{n}\sum_i f(\mathbf{x}^i)$ denotes the sample average and $q(\mathbf{z} \mid \mathbf{x})$ is non-negative and sums to 1 over $\mathbf{z}$ for each $\mathbf{x}$. The lower bound above is a simple consequence of Jensen's inequality for the $\log$ function. It can be shown that the lower bound can be made tight for a given value of $\theta$ by maximizing over $q$ and under mild continuity conditions on $p_\theta(\mathbf{x}, \mathbf{z})$, local maxima $(q^*, \theta^*)$ of $F(q, \theta)$ correspond to local maxima $\theta^*$ of $L_S(\theta)$ [14].

Standard EM iteration performs coordinate ascent on $F(q, \theta)$ as follows:

$$
\mathbf{E}: \quad q^{t+1}(\mathbf{z} \mid \mathbf{x}) = \underset{q(\mathbf{z}|\mathbf{x})}{\arg\max} F(q, \theta^t) = \underset{q(\mathbf{z}|\mathbf{x})}{\arg\min} \mathrm{KL}(q(\mathbf{z} \mid \mathbf{x}) \,\|\, p_{\theta^t}(\mathbf{z} \mid \mathbf{x})) = p_{\theta^t}(\mathbf{z} \mid \mathbf{x}); \quad (3)
$$

$$
\mathbf{M}: \quad \theta^{t+1} = \underset{\theta}{\arg\max} F(q^{t+1}, \theta) = \underset{\theta}{\arg\max} \mathbf{E}_S\left[\sum_{\mathbf{z}} q^{t+1}(\mathbf{z} \mid \mathbf{x}) \log p_\theta(\mathbf{x}, \mathbf{z})\right], \quad (4)
$$

where $\mathrm{KL}(q\|p) = \mathbf{E}_q[\log \frac{q(\cdot)}{p(\cdot)}]$ is Kullback-Leibler divergence. The E step computes the posteriors of the latent variables given the observed variables and current parameters. The M step uses $q$ to "fill in" the values of latent variables $\mathbf{z}$ and estimate parameters $\theta$ as if the data was complete. This step is particularly easy for exponential models, where $\theta$ is a simple function of the (expected) sufficient statistics. This modular split into two intuitive and straightforward steps accounts for the vast popularity of EM. In the following, we build on this simple scheme while incorporating desired constraints on the posteriors over latent variables.

### 2.1 Constraining the posteriors

Our goal is to allow for finer-level control over posteriors, bypassing the likelihood function. We propose an intuitive way to modify EM to accomplish this and discuss the implications of the new procedure below in terms of the objective it attempts to optimize. We can express our desired constraints on the posteriors as the requirement that $p_\theta(\mathbf{z} \mid \mathbf{x}) \in \mathcal{Q}(\mathbf{x})$. For example, in dependency grammar induction, constraining the average length of dependency attachments is desired [17]; in statistical word alignment, the constraint might involve the expected degree of each node in the alignment [3]. Instead of restricting $p$ directly, which might not be feasible, we can penalize the distance of $p$ to the constraint set $\mathcal{Q}$. As it turns out, we can accomplish this by restricting $q$ to be constrained to $\mathcal{Q}$ instead. This results in a very simple modification to the E step of EM, by constraining the set of $q$ over which $F(q, \theta)$ is optimized (M step is unchanged):

$$
\mathbf{E}: \quad q^{t+1}(\mathbf{z} \mid \mathbf{x}) = \underset{q(\mathbf{z}|\mathbf{x}) \in \mathcal{Q}(\mathbf{x})}{\arg\max} F(q, \theta^t) = \underset{q(\mathbf{z}|\mathbf{x}) \in \mathcal{Q}(\mathbf{x})}{\arg\min} \mathrm{KL}(q(\mathbf{z} \mid \mathbf{x}) \,\|\, p_{\theta^t}(\mathbf{z} \mid \mathbf{x})) \quad (5)
$$

Note that in variational EM, the set $\mathcal{Q}(x)$ is usually a simpler inner bound (as in mean field) or outer bound (as in loopy belief propagation) on the intractable original space of posteriors [9]. The situation here is the opposite: we assume the original posterior space is tractable but we add constraints to enforce intended semantics not captured by the simple model. Of course to make this practical, the set $\mathcal{Q}(\mathbf{x})$ needs to be well-behaved. We assume that $\mathcal{Q}(\mathbf{x})$ is convex and non-empty for every $\mathbf{x}$ so that the problem in Eq. (5) becomes a strictly convex minimization over a non-empty convex set, guaranteed to have a unique minimizer [1]. A natural and general way to specify constraints on $q$ is by bounding expectations of given functions: $\mathbf{E}_q[f(\mathbf{x}, \mathbf{z})] \leq b$ (equality can be achieved by adding $\mathbf{E}_q[-f(\mathbf{x}, \mathbf{z})] \leq -b$). Stacking functions $f()$ into a vector $\mathbf{f}()$ and constants $b$ into a vector $\mathbf{b}$, the minimization problem in Eq. (5) becomes:

$$\arg\min_q \mathrm{KL}(q(\mathbf{z} \mid \mathbf{x}) \,\|\, p_{\theta^t}(\mathbf{z} \mid \mathbf{x})) \ \text{ s.t. } \ \mathbf{E}_q[\mathbf{f}(\mathbf{x}, \mathbf{z})] \leq \mathbf{b}. \tag{6}$$

In the next section, we discuss how to solve this optimization problem (also called I-projection in information geometry), but before we move on, it is interesting to consider what this new procedure in Eq. (5) converges to. The new scheme alternately maximizes $F(q, \theta)$, but over a subspace of the original space of $q$, hence using a looser lower-bound than original EM. We are no longer guaranteed that the local maxima of the constrained problem are local maxima of the log-likelihood. However, we can characterize the objective maximized at local maxima as log-likelihood penalized by average KL divergence of posteriors from $\mathcal{Q}$:

**Proposition 2.1** *The local maxima of $F(q, \theta)$ such that $q(\mathbf{z} \mid \mathbf{x}) \in \mathcal{Q}(\mathbf{x}), \forall \mathbf{x} \in S$ are local maxima of*

$$\mathbf{E}_S[\log p_\theta(\mathbf{x})] - \mathbf{E}_S[\mathrm{KL}(\mathcal{Q}(\mathbf{x}) \,\|\, p_\theta(\mathbf{z} \mid \mathbf{x})],$$

*where* $\mathrm{KL}(\mathcal{Q}(\mathbf{x}) \,\|\, p_\theta(\mathbf{z} \mid \mathbf{x}) = \min_{q(\mathbf{z}|\mathbf{x})) \in \mathcal{Q}(\mathbf{x})} \mathrm{KL}(q(\mathbf{z} \mid \mathbf{x}) \,\|\, p_\theta(\mathbf{z} \mid \mathbf{x})).$

**Proof:** By adding and subtracting $\mathbf{E}_S[\sum_\mathbf{z} q(\mathbf{z} \mid \mathbf{x}) \log p_\theta(\mathbf{z} \mid \mathbf{x})]$ from $F(q, \theta)$, we get:

$$F(q, \theta) = \mathbf{E}_S\left[\sum_\mathbf{z} q(\mathbf{z} \mid \mathbf{x}) \log \frac{p_\theta(\mathbf{x}, \mathbf{z})}{q(\mathbf{z} \mid \mathbf{x})}\right] \tag{7}$$

$$= \mathbf{E}_S\left[\sum_\mathbf{z} q(\mathbf{z} \mid \mathbf{x}) \log \frac{p_\theta(\mathbf{x}, \mathbf{z})}{p_\theta(\mathbf{z} \mid \mathbf{x})}\right] - \mathbf{E}_S\left[\sum_\mathbf{z} q(\mathbf{z} \mid \mathbf{x}) \log \frac{q(\mathbf{z} \mid \mathbf{x})}{p_\theta(\mathbf{z} \mid \mathbf{x})}\right] \tag{8}$$

$$= \mathbf{E}_S\left[\sum_\mathbf{z} q(\mathbf{z} \mid \mathbf{x}) \log p_\theta(\mathbf{x})\right] - \mathbf{E}_S[\mathrm{KL}(q(\mathbf{z} \mid \mathbf{x}) \| p_\theta(\mathbf{z} \mid \mathbf{x})] \tag{9}$$

$$= \mathbf{E}_S[\log p_\theta(\mathbf{x})] - \mathbf{E}_S[\mathrm{KL}(q(\mathbf{z} \mid \mathbf{x}) \,\|\, p_\theta(\mathbf{z} \mid \mathbf{x})]. \tag{10}$$

Since the first term does not depend on $q$, the second term is minimized by $q^*(\mathbf{z} \mid \mathbf{x}) = \min_{q(\mathbf{z}|\mathbf{x})) \in \mathcal{Q}(\mathbf{x})} \mathrm{KL}(q(\mathbf{z} \mid \mathbf{x}) \,\|\, p_\theta(\mathbf{z} \mid \mathbf{x}))$ at local maxima.

This proposition implies that our procedure trades off likelihood and distance to the desired posterior subspace (modulo getting stuck in local maxima) and provides an effective method of controlling the posteriors.

## 2.2 Computing I-projections onto $\mathcal{Q}(\mathbf{x})$

The KL-projection onto $\mathcal{Q}(\mathbf{x})$ in Eq. (6) is easily solved via the dual (cf. [5, 1]):

$$\arg\max_{\lambda \geq 0} \left( \lambda^\top \mathbf{b} - \log \sum_\mathbf{z} p_{\theta^t}(\mathbf{z} \mid \mathbf{x}) \exp\{\lambda^\top \mathbf{f}(\mathbf{x}, \mathbf{z})\} \right) \tag{11}$$

Define $q_\lambda(\mathbf{z} \mid \mathbf{x}) \propto p_{\theta^t}(\mathbf{z} \mid \mathbf{x}) \exp\{\lambda^\top \mathbf{f}(\mathbf{x}, \mathbf{z})\}$, then at the dual optimum $\lambda^*$, the primal solution is given by $q_{\lambda^*}(\mathbf{z} \mid \mathbf{x})$.

Such projections become particularly efficient when we assume the constraint functions decompose the same way as the graphical model: $f(\mathbf{x}, \mathbf{z}) = \sum_\alpha f(\mathbf{x}_\alpha, \mathbf{z}_\alpha)$. Then $q_\lambda(\mathbf{z} \mid \mathbf{x}) \propto \prod_\alpha \phi_{\theta^t}(\mathbf{x}_\alpha, \mathbf{z}_\alpha) \exp\{\lambda^\top \mathbf{f}(\mathbf{x}_\alpha, \mathbf{z}_\alpha)\}$, which factorizes the same way as $p_\theta(\mathbf{x}, \mathbf{z})$. In case the constraint functions do not decompose over the model cliques but require additional cliques, the resulting $q_\lambda$ will factorize over the union of the original cliques and the constraint function cliques,

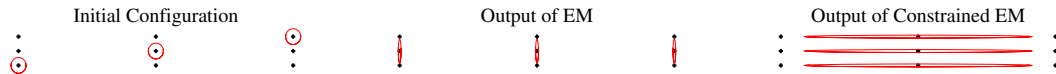

Figure 1: Synthetic data results. The dataset consists of 9 points drawn as dots and there are three clusters represented by ovals centered at their mean with dimensions proportional to their standard deviation. The EM algorithm clusters each column of points together, but if we introduce the constraint that each column should have at least one of the clusters, we get the clustering to the right.

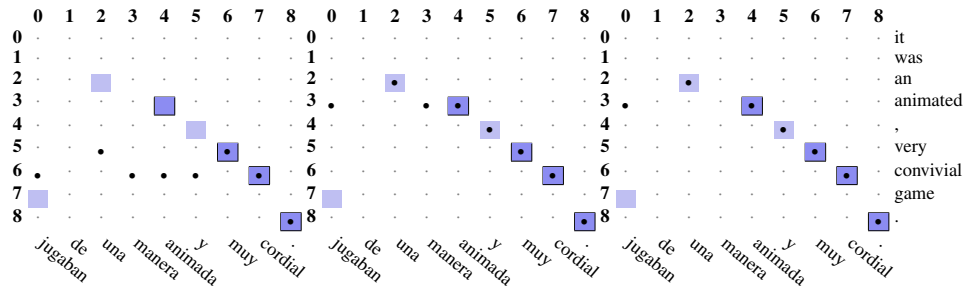

Figure 2: An example of the output of HMM trained on 100k the EPPS data. Left: Baseline model. Middle: Substochastic constraints. Right Agreement constraints.

potentially making inference more expensive. In our experiments, we used constraint functions that decompose with the original model. Note that even in this case, the graphical model $p_\theta(\mathbf{x}, \mathbf{z})$ can not in general satisfy the expectation constraints for every setting of $\theta$ and $\mathbf{x}$. Instead, the constrained EM procedure is tuning $\theta$ to the distribution of $\mathbf{x}$ to satisfy these constraints in expectation.

The dual of the projection problem can be solved using a variety of optimization methods; perhaps the simplest of them is projected gradient (since $\lambda$ is non-negative, we need to simply truncate negative values as we perform gradient ascent). The gradient of the objective in Eq. (11) is given by: $\mathbf{b} - \mathbf{E}_{q_\lambda}[\mathbf{f}(\mathbf{x}, \mathbf{z})] = \mathbf{b} - \sum_\alpha \mathbf{E}_{q_\lambda(\mathbf{z}_\alpha | \mathbf{x})}[\mathbf{f}(\mathbf{x}_\alpha, \mathbf{z}_\alpha)]$. Every gradient computation thus involves computing marginals of $q_\lambda(\mathbf{z} \mid \mathbf{x})$, which is of the same complexity as computing marginals of $p_\theta(\mathbf{z} \mid \mathbf{x})$ if no new cliques are added by the constraint functions. In practice, we do not need to solve the dual to a very high precision in every round of EM, so several (about 5-10) gradient steps suffice. When the number of constraints is small, alternating projections are also a good option.

## 3 Clustering

A simple but common problem that employs EM is clustering a group of points using a mixture of Gaussians. In practice, the data points and Gaussian clusters have some meaning not captured by the model. For example, the data points could correspond to descriptors of image parts and the clusters could be "words" used for later processing of the image. In that case, we often have special knowledge about the clusters that we expect to see that is difficult to express in the original model. For example, we might know that within each image two features that are of different scales should not be clustered together. As another example, we might know that each image has at least one copy of each cluster. Both of these constraints are easy to capture and implement in our framework. Let $z_{ij} = 1$ represent the event that data point $i$ is assigned to cluster $j$. If we want to ensure that data point $i$ is not assigned to the same cluster as data point $i'$ then we need to enforce the constraint $\mathbf{E}[z_{ij} + z_{i'j}] \le 1, \forall j$. To ensure the constraint that each cluster has at least one data point assigned to it from an instance $I$ we need to enforce the constraint $\mathbf{E}\left[\sum_{i \in I} z_{ij}\right] \le 1, \forall j$. We implemented this constraint in a mixture of Gaussians clustering algorithm. Figure 1 compares clustering of synthetic data using unconstrained EM as well as our method with the constraint that each column of data points has at least one copy of each cluster in expectation.

# 4 Statistical word alignment

Statistical word alignment, used primarily for machine translation, is a task where the latent variables are intended to have a meaning: whether a word in one language translates into a word in another language in the context of the given sentence pair. The input to an alignment systems is a sentence aligned bilingual corpus, consisting of pairs of sentences in two languages. Figure 2 shows three machine-generated alignments of a sentence pair. The black dots represent the machine alignments and the shading represents the human annotation. Darkly shaded squares with a border represent a *sure* alignments that the system is required to produce while lightly shaded squares without a border represent *possible* alignments that the system is optionally allowed to produce.

We denote one language the "source" language and use $\mathbf{s}$ for its sentences and one language the "target" language and use $\mathbf{t}$ for its sentences. It will also be useful to talk about an alignment for a particular sentence pair as a binary matrix $\mathbf{z}$, with $z_{ij} = 1$ representing "source word $i$ generates target word $j$." The generative models we consider generate target word $j$ from only one source word, and so an alignment is only valid from the point of view of the model when $\sum_i z_{ij} = 1$, so we can equivalently represent an alignment as an array $a$ of indices, with $a_j = i \Leftrightarrow z_{ij} = 1$.

Figure 2 shows three alignments performed by a baseline model as well as our two modifications. We see that the rare word "convivial" acts as a garbage collector[2], aligning to words that do not have a simple translation in the target sentence. Both of the constraints we suggest repair this problem to different degrees. We now introduce the baseline models and the constraints we impose on them.

## 4.1 Baseline models

We consider three models below: IBM Model 1, IBM Model 2 [3] and the HMM model proposed by [20]. The three models can be expressed as:

$$p(\mathbf{t}, \mathbf{a} \mid \mathbf{s}) = \prod_j p_d(a_j | j, a_{j-1}) p_t(\mathbf{t}_j | \mathbf{s}_{a_j}), \tag{12}$$

with the three models differing in their definition of the distortion probability $p_d(a_j | j, a_{j-1})$. Model 1 assumes that the positions of the words are not important and assigns uniform distortion probability. Model 2 allows a dependence on the positions $p_d(a_j | j, a_{j-1}) = p_d(a_j | j)$ and the HMM model assumes that the only the distance between the current and previous source word are important $p_d(a_j | j, a_{j-1}) = p_d(a_j | a_j - a_{j-1})$. All the models are augmented by adding a special "null" word to the source sentence. The likelihood of the corpus, marginalized over possible alignments is concave for Model 1, but not for the other models [3].

### 4.1.1 Substochastic Constraints

A common error for our baseline models is to use rare source words as garbage collectors [2]. The models align target words that do not match any of the source words to rare source words rather than to the null word. While this results in higher data likelihood, the resulting alignments are not desirable, since they cannot be interpreted as translations. Figure 2 shows an example. One might consider augmenting the models to disallow this, for example by restricting that the alignments are at most one-to-one. Unfortunately computing the normalization for such a model is a ♯P complete problem [19]. Our approach is to instead constrain the posterior distribution over alignments during the E-step. More concretely we enforce the constraint $\mathbf{E}_q[z_{ij}] \leq 1$. Another way of thinking of this constraint is that we require the expected fertility of each source word to be at most one. For our hand-aligned corpora Hansards [15] and EPPS [11, 10], the average fertility is around $1$ and $1.2$, respectively, with standard deviation of $0.01$. We will see that these constraints improve alignment accuracy.

### 4.1.2 Agreement Constraints

Another weakness of our baseline models is that they are asymmetric. Usually, a model is trained in each direction and then they are heuristically combined. [12] introduce an objective to train the two models concurrently and encourage them to agree. Unfortunately their objective leads to an intractable E-step and they are forced to use a heuristic approximation. In our framework, we can

| Hansards 447 sentences | | | | | EPPS 400 sentences | | | |
|---|---|---|---|---|---|---|---|---|
| Language | Max | Avg. | Fertility | Avg. F. | Language | Max | Avg. | Fertility | Avg. F. |
| English | 30 | 15.7 | 6 | 1.02 | English | 90 | 29 | 218 | 1.20 |
| French | 30 | 17.4 | 3 | 1.00 | Spanish | 99 | 31.2 | 165 | 1.17 |

Table 1: Test Corpus statistics. Max and Avg. refer to sentence length. Fertility is the number of words that occur at least twice and have on average at least 1.5 sure alignment when they have any. Avg. F. is the average word fertility. All average fertilities have a standard deviation of 0.01.

also enforce agreement in expectation without approximating. Denote one direction the "forward" direction and the other the "backward" direction. Denote the forward model $\overrightarrow{p}$ with hidden variables $\overrightarrow{\mathbf{z}} \in \overrightarrow{\mathbf{Z}}$ and backward model $\overleftarrow{p}$ with hidden variables $\overleftarrow{\mathbf{z}} \in \overleftarrow{\mathbf{Z}}$ and note $\overrightarrow{p}(\overleftarrow{\mathbf{z}}) = 0$ and $\overleftarrow{p}(\overrightarrow{\mathbf{z}}) = 0$. Define a mixture $p(\mathbf{z}) = \frac{1}{2}\overrightarrow{p}(\mathbf{z}) + \frac{1}{2}\overleftarrow{p}(\mathbf{z})$ for $\mathbf{z} \in \overleftarrow{\mathbf{Z}} \cup \overrightarrow{\mathbf{Z}}$. The constraints that enforce agreement in this setup are $\mathbf{E}_q[\mathbf{f}(\mathbf{x}, \mathbf{z})] = \mathbf{0}$ with

$$
f_{ij}(\mathbf{x}, \mathbf{z}) = \begin{cases} 1 & \mathbf{z} \in \overrightarrow{\mathbf{Z}} \text{ and } z_{ij} = 1 \\ -1 & \mathbf{z} \in \overleftarrow{\mathbf{Z}} \text{ and } z_{ij} = 1 \\ 0 & \text{otherwise} \end{cases} .
$$

## 5   Evaluation

We evaluated our augmented models on two corpora: the Hansards corpus [15] of English/French and the Europarl corpus [10] with EPPS annotation [11]. Table 1 presents some statistics for the two corpora. Notably, Hansards is a much easier corpus than EPPS. Hansards test sentences are on average only half as long as those of EPPS and only 21% of alignments in Hansards are *sure* and hence required compared with 69% for EPPS. Additionally, more words in EPPS are aligned to multiple words in the other language. Since our models cannot model this "fertility" we expect their performance to be worse on EPPS data. Despite these differences, the corpora are also similar in some ways. Both are alignments of a Romance language to English and the average distance of an alignment to the diagonal is around 2 for both corpora.

The error metrics we use are precision, recall and alignment error rate (AER), which is a weighted combination of precision and recall. Although AER is the standard metric in word alignment is has been shown [7] that it has a weak correlation with the standard MT metric, Bleu, when the alignments are used in a phrase-based translation system. [7] suggest weighted F-Measure[1] as an alternative that correlates well with Bleu, so we also report precision and recall numbers.

Following prior work [16], we initialize Model 1 translation table with uniform probabilities over word pairs that occur together in same sentence. Model 2 and Model HMM were initialized with the translation probabilities from Model 1 and with uniform distortion probabilities. All models were trained for 5 iterations. We used a maximum length cutoff for training sentences of 40. For the Hansards corpus this leaves 87.3% of the sentences, while for EPPS this leaves 74.5%. Following common practice, we included the unlabeled test and development data during training. We report results for the model with English as the "source" language when using posterior decoding [12].

Figures 3 shows alignment results for the baselines models as well as the models with additional constraints. We show precision, recall and AER for the HMM model as well as precision and recall for Model 2. We note that both constraints improve all measures of performance for all dataset sizes, with most improvement for smaller dataset sizes.

We performed additional experiments to verify that our model is not unfairly aided by the standard but arbitrary choice of 5 iterations of EM. Figure 4 shows AER and data likelihood as a function of the number of EM iterations. We see that the performance gap between the model with and without agreement constraints is preserved as the number of EM iterations increases. Note also that likelihood increases monotonically for all the models and that the baseline model always achieves higher likelihood as expected.

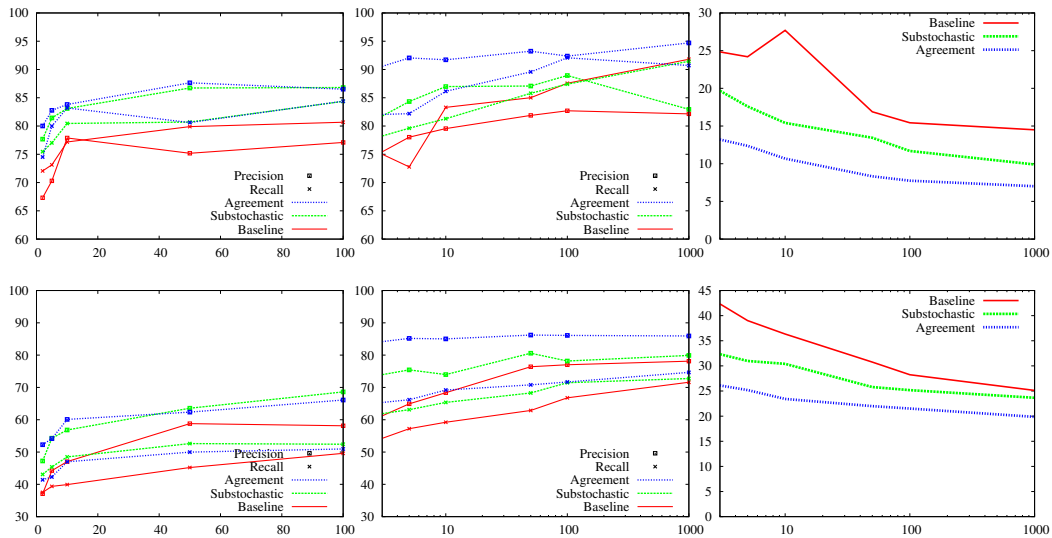

Figure 3: Effect of posterior constraints on learning curves for IBM Model 2 and HMM. From left to right: Precision/Recall for IBM Model 2, Precision/Recall for HMM Model and AER for HMM Model. Top: Hansards Bottom: EPPS. Both types of constraints improve all accuracy measures across both datasets and models.

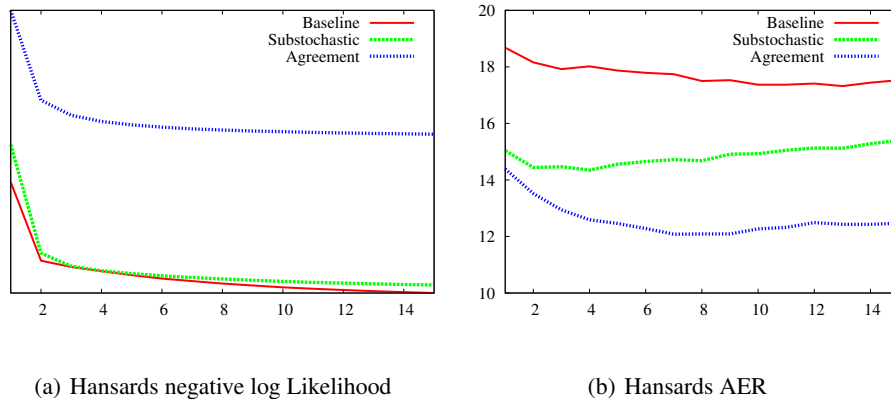

(a) Hansards negative log Likelihood        (b) Hansards AER

Figure 4: Data likelihood and AER vs. EM iteration using HMM on 100k Hansards.

# 6 Conclusions

In this paper we described a general and principled way to introduce prior knowledge to guide the EM algorithm. Intuitively, we can view our method as a way to exert flexible control during the execution of EM. More formally, our method can be viewed as a regularization of the expectations of the hidden variables during EM. Alternatively, it can be viewed as an augmentation of the EM objective function with KL divergence from a set of feasible models. We implemented our method on two different problems: probabilistic clustering using mixtures of Gaussians and statistical word alignment and tested it on synthetic and real data. We observed improved performance by introducing simple and intuitive prior knowledge into the learning process. Our method is widely applicable to other problems where the EM algorithm is used but prior knowledge about the problem is hard to introduce directly into the model.

**Acknowledgments**

J. V. Graça was supported by a fellowship from Fundação para a Ciência e Tecnologia (SFRH/ BD/ 27528/ 2006). K. Ganchev was partially supported by NSF ITR EIA 0205448.

## Footnotes

[1]defined as $\left(\frac{\alpha}{Precision} + \frac{1-\alpha}{Recall}\right)^{-1}$ with $0.1 \leq \alpha \leq 0.4$ showing good correlation with Bleu [7].

## References

[1] D. Bertsekas. *Nonlinear Programming*. Athena Scientific, Belmont, MA, 1999.

[2] P. F. Brown, S. A. Della Pietra, V. J. Della Pietra, M. J. Goldsmith, J. Hajic, R. L. Mercer, and S. Mohanty. But dictionaries are data too. In *Proc. HLT*, 1993.

[3] Peter F. Brown, Stephen Della Pietra, Vincent J. Della Pietra, and Robert L. Mercer. The mathematic of statistical machine translation: Parameter estimation. *Computational Linguistics*, 19(2):263–311, 1994.

[4] O. Chapelle, B. Schölkopf, and A. Zien, editors. *Semi-Supervised Learning*. MIT Press, Cambridge, MA, 2006.

[5] I. Csiszar. I-divergence geometry of probability distributions and minimization problems. *The Annals of Probability*, 3, 1975.

[6] A. P. Dempster, N. M. Laird, and D. B. Rubin. Maximum likelihood from incomplete data via the em algorithm. *Journal of the Royal Statistical Society. Series B (Methodological)*, 39(1):1–38, 1977.

[7] Alexander Fraser and Daniel Marcu. Measuring word alignment quality for statistical machine translation. *Comput. Linguist.*, 33(3):293–303, 2007.

[8] Nir Friedman, Ori Mosenzon, Noam Slonim, and Naftali Tishby. Multivariate information bottleneck. In *UAI*, 2001.

[9] Michael I. Jordan, Zoubin Ghahramani, Tommi Jaakkola, and Lawrence K. Saul. An introduction to variational methods for graphical models. *Machine Learning*, 37(2):183–233, 1999.

[10] Philipp Koehn. Europarl: A multilingual corpus for evaluation of machine translation, 2002.

[11] P. Lambert, A.De Gispert, R. Banchs, and J. B. Mariño. Guidelines for word alignment evaluation and manual alignment. In *Language Resources and Evaluation, Volume 39, Number 4*, pages 267–285, 2005.

[12] Percy Liang, Ben Taskar, and Dan Klein. Alignment by agreement. In *Proc. HLT-NAACL*, 2006.

[13] Gideon S. Mann and Andrew McCallum. Simple, robust, scalable semi-supervised learning via expectation regularization. In *Proc. ICML*, 2007.

[14] R. M. Neal and G. E. Hinton. A new view of the EM algorithm that justifies incremental, sparse and other variants. In M. I. Jordan, editor, *Learning in Graphical Models*, pages 355–368. Kluwer, 1998.

[15] Franz Josef Och and Hermann Ney. Improved statistical alignment models. In *ACL*, 2000.

[16] Franz Josef Och and Hermann Ney. A systematic comparison of various statistical alignment models. *Comput. Linguist.*, 29(1):19–51, 2003.

[17] Noah A. Smith and Jason Eisner. Annealing structural bias in multilingual weighted grammar induction. In *Proc. ACL*, pages 569–576, 2006.

[18] Martin Szummer and Tommi Jaakkola. Information regularization with partially labeled data. In *Proc. NIPS*, pages 1025–1032, 2003.

[19] L. G. Valiant. The complexity of computing the permanent. *Theoretical Computer Science*, 8:189–201, 1979.

[20] Stephan Vogel, Hermann Ney, and Christoph Tillmann. Hmm-based word alignment in statistical translation. In *Proc. COLING*, 1996.

